# Minimax Probability Machine

**Gert R.G. Lanckriet**[*]
Department of EECS
University of California, Berkeley
Berkeley, CA 94720-1770
*gert@eecs.berkeley.edu*

**Laurent El Ghaoui**
Department of EECS
University of California, Berkeley
Berkeley, CA 94720-1770
*elghaoui@eecs.berkeley.edu*

**Chiranjib Bhattacharyya**
Department of EECS
University of California, Berkeley
Berkeley, CA 94720-1776
*chiru@eecs.berkeley.edu*

**Michael I. Jordan**
Computer Science and Statistics
University of California, Berkeley
Berkeley, CA 94720-1776
*jordan@cs.berkeley.edu*

## Abstract

When constructing a classifier, the probability of correct classification of future data points should be maximized. In the current paper this desideratum is translated in a very direct way into an optimization problem, which is solved using methods from convex optimization. We also show how to exploit Mercer kernels in this setting to obtain nonlinear decision boundaries. A worst-case bound on the probability of misclassification of future data is obtained explicitly.

## 1   Introduction

Consider the problem of choosing a linear discriminant by minimizing the probabilities that data vectors fall on the wrong side of the boundary. One way to attempt to achieve this is via a generative approach in which one makes distributional assumptions about the class-conditional densities and thereby estimates and controls the relevant probabilities. The need to make distributional assumptions, however, casts doubt on the generality and validity of such an approach, and in discriminative solutions to classification problems it is common to attempt to dispense with class-conditional densities entirely.

Rather than avoiding any reference to class-conditional densities, it might be useful to attempt to control misclassification probabilities in a worst-case setting; that is, under all possible choices of class-conditional densities. Such a minimax approach could be viewed as providing an alternative justification for discriminative approaches. In this paper we show how such a minimax programme can be carried out in the setting of binary classification. Our approach involves exploiting the following powerful theorem due to Isii [6], as extended in recent work by Bertsimas

---

[*] http://robotics.eecs.berkeley.edu/˜gert/

and Sethuraman [2]:

$$\sup \Pr\{\mathbf{a}^T\mathbf{y} \geq b\} = \frac{1}{1+d^2} \ , \ \text{with} \quad d^2 = \inf_{\mathbf{a}^T\mathbf{y} \geq b} (\mathbf{y} - \bar{\mathbf{y}})^T \boldsymbol{\Sigma_\mathbf{y}}^{-1}(\mathbf{y} - \bar{\mathbf{y}}), \quad (1)$$

where $\mathbf{y}$ is a random vector, where $\mathbf{a}$ and $b$ are constants, and where the supremum is taken over all distributions having mean $\bar{\mathbf{y}}$ and covariance matrix $\boldsymbol{\Sigma_\mathbf{y}}$. This theorem provides us with the ability to bound the probability of misclassifying a point, without making Gaussian or other specific distributional assumptions. We will show how to exploit this ability in the design of linear classifiers.

One of the appealing features of this formulation is that one obtains an explicit upper bound on the probability of misclassification of future data: $1/(1 + d^2)$.

A second appealing feature of this approach is that, as in linear discriminant analysis [7], it is possible to generalize the basic methodology, utilizing Mercer kernels and thereby forming nonlinear decision boundaries. We show how to do this in Section 3.

The paper is organized as follows: in Section 2 we present the minimax formulation for linear classifiers, while in Section 3 we deal with kernelizing the method. We present empirical results in Section 4.

## 2  Maximum probabilistic decision hyperplane

In this section we present our minimax formulation for linear decision boundaries. Let $\mathbf{x}$ and $\mathbf{y}$ denote random vectors in a binary classification problem, with mean vectors and covariance matrices given by $\mathbf{x} \sim (\bar{\mathbf{x}}, \boldsymbol{\Sigma_\mathbf{x}})$ and $\mathbf{y} \sim (\bar{\mathbf{y}}, \boldsymbol{\Sigma_\mathbf{y}})$, respectively, where "$\sim$" means that the random variable has the specified mean and covariance matrix but that the distribution is otherwise unconstrained. Note that $\mathbf{x}, \bar{\mathbf{x}}, \mathbf{y}, \bar{\mathbf{y}} \in \mathbb{R}^n$ and $\boldsymbol{\Sigma_\mathbf{x}}, \boldsymbol{\Sigma_\mathbf{y}} \in \mathbb{R}^{n \times n}$.

We want to determine the hyperplane $\mathbf{a}^T\mathbf{z} = b$ ($\mathbf{a}, \mathbf{z} \in \mathbb{R}^n$ and $b \in \mathbb{R}$) that separates the two classes of points with maximal probability with respect to all distributions having these means and covariance matrices. This boils down to:

$$\max_{\alpha,\mathbf{a},b} \alpha \quad \text{s.t.} \quad \inf \Pr\{\mathbf{a}^T\mathbf{x} \geq b\} \geq \alpha \tag{2}$$

$$\inf \Pr\{\mathbf{a}^T\mathbf{y} \leq b\} \geq \alpha$$

or,

$$\max_{\alpha,\mathbf{a},b} \alpha \quad \text{s.t.} \quad 1 - \alpha \geq \sup \Pr\{\mathbf{a}^T\mathbf{x} \leq b\} \tag{3}$$

$$1 - \alpha \geq \sup \Pr\{\mathbf{a}^T\mathbf{y} \geq b\}.$$

Consider the second constraint in (3). Recall the result of Bertsimas and Sethuraman [2]:

$$\sup \Pr\{\mathbf{a}^T\mathbf{y} \geq b\} = \frac{1}{1+d^2} \ , \ \text{with} \quad d^2 = \inf_{\mathbf{a}^T\mathbf{y} \geq b} (\mathbf{y} - \bar{\mathbf{y}})^T \boldsymbol{\Sigma_\mathbf{y}}^{-1}(\mathbf{y} - \bar{\mathbf{y}}) \tag{4}$$

We can write this as $d^2 = \inf_{\mathbf{c}^T\mathbf{w} \geq d} \mathbf{w}^T\mathbf{w}$, where $\mathbf{w} = \boldsymbol{\Sigma_\mathbf{y}}^{-1/2}(\mathbf{y} - \bar{\mathbf{y}})$, $\mathbf{c}^T = \mathbf{a}^T \boldsymbol{\Sigma_\mathbf{y}}^{1/2}$ and $d = b - \mathbf{a}^T\bar{\mathbf{y}}$. To solve this, first notice that we can assume that $\mathbf{a}^T\bar{\mathbf{y}} \leq b$ (i.e. $\bar{\mathbf{y}}$ is classified correctly by the decision hyperplane $\mathbf{a}^T\mathbf{z} = b$): indeed, otherwise we would find $d^2 = 0$ and thus $\alpha = 0$ for that particular $\mathbf{a}$ and $b$, which can never be an optimal value. So, $d > 0$. We then form the Lagrangian:

$$\mathcal{L}(\mathbf{w}, \lambda) = \mathbf{w}^T\mathbf{w} + \lambda(d - \mathbf{c}^T\mathbf{w}), \tag{5}$$

which is to be maximized with respect to $\lambda \geq 0$ and minimized with respect to $\mathbf{w}$. At the optimum, $2\mathbf{w} = \lambda\mathbf{c}$ and $d = \mathbf{c}^T\mathbf{w}$, so $\lambda = \frac{2d}{\mathbf{c}^T\mathbf{c}}$ and $\mathbf{w} = \frac{d\mathbf{c}}{\mathbf{c}^T\mathbf{c}}$. This yields:

$$d^2 = \inf_{\mathbf{a}^T\mathbf{y}\geq b} (\mathbf{y}-\bar{\mathbf{y}})^T \boldsymbol{\Sigma_y}^{-1}(\mathbf{y}-\bar{\mathbf{y}}) = \frac{(b-\mathbf{a}^T\bar{\mathbf{y}})^2}{\mathbf{a}^T\boldsymbol{\Sigma_y}\mathbf{a}} \tag{6}$$

Using (4), the second constraint in (3) becomes $1-\alpha \geq 1/(1+d^2)$ or $d^2 \geq \alpha/(1-\alpha)$. Taking (6) into account, this boils down to:

$$b - \mathbf{a}^T\bar{\mathbf{y}} \geq \kappa(\alpha)\sqrt{\mathbf{a}^T\boldsymbol{\Sigma_y}\mathbf{a}} \quad \text{where} \quad \kappa(\alpha) = \sqrt{\frac{\alpha}{1-\alpha}} \tag{7}$$

We can handle the first constraint in (3) in a similar way (just write $\mathbf{a}^T\mathbf{x} \leq b$ as $-\mathbf{a}^T\mathbf{x} \geq -b$ and apply the result (7) for the second constraint). The optimization problem (3) then becomes:

$$\max_{\alpha,\mathbf{a},b} \alpha \quad \text{s.t.} \quad -b+\mathbf{a}^T\bar{\mathbf{x}} \geq \kappa(\alpha)\sqrt{\mathbf{a}^T\boldsymbol{\Sigma_x}\mathbf{a}} \tag{8}$$

$$b - \mathbf{a}^T\bar{\mathbf{y}} \geq \kappa(\alpha)\sqrt{\mathbf{a}^T\boldsymbol{\Sigma_y}\mathbf{a}}.$$

Because $\kappa(\alpha)$ is a monotone increasing function of $\alpha$, we can write this as:

$$\max_{\kappa,\mathbf{a},b} \kappa \quad \text{s.t.} \quad -b+\mathbf{a}^T\bar{\mathbf{x}} \geq \kappa\sqrt{\mathbf{a}^T\boldsymbol{\Sigma_x}\mathbf{a}} \tag{9}$$

$$b - \mathbf{a}^T\bar{\mathbf{y}} \geq \kappa\sqrt{\mathbf{a}^T\boldsymbol{\Sigma_y}\mathbf{a}}.$$

From both constraints in (9), we get

$$\mathbf{a}^T\bar{\mathbf{y}} + \kappa\sqrt{\mathbf{a}^T\boldsymbol{\Sigma_y}\mathbf{a}} \leq b \leq \mathbf{a}^T\bar{\mathbf{x}} - \kappa\sqrt{\mathbf{a}^T\boldsymbol{\Sigma_x}\mathbf{a}}, \tag{10}$$

which allows us to eliminate $b$ from (9):

$$\max_{\kappa,\mathbf{a}} \kappa \quad \text{s.t.} \quad \mathbf{a}^T\bar{\mathbf{y}} + \kappa\sqrt{\mathbf{a}^T\boldsymbol{\Sigma_y}\mathbf{a}} \leq \mathbf{a}^T\bar{\mathbf{x}} - \kappa\sqrt{\mathbf{a}^T\boldsymbol{\Sigma_x}\mathbf{a}}. \tag{11}$$

Because we want to maximize $\kappa$, it is obvious that the inequalities in (10) will become equalities at the optimum. The optimal value of $b$ will thus be given by

$$b_* = \mathbf{a}_*^T\bar{\mathbf{x}} - \kappa_*\sqrt{\mathbf{a}_*^T\boldsymbol{\Sigma_x}\mathbf{a}_*} = \mathbf{a}_*^T\bar{\mathbf{y}} + \kappa_*\sqrt{\mathbf{a}_*^T\boldsymbol{\Sigma_y}\mathbf{a}_*}. \tag{12}$$

where $\mathbf{a}_*$ and $\kappa_*$ are the optimal values of $\mathbf{a}$ and $\kappa$ respectively. Rearranging the constraint in (11), we get:

$$\mathbf{a}^T(\bar{\mathbf{x}} - \bar{\mathbf{y}}) \geq \kappa\left(\sqrt{\mathbf{a}^T\boldsymbol{\Sigma_x}\mathbf{a}} + \sqrt{\mathbf{a}^T\boldsymbol{\Sigma_y}\mathbf{a}}\right). \tag{13}$$

The above is positively homogeneous in $\mathbf{a}$: if $\mathbf{a}$ satisfies (13), $s\mathbf{a}$ with $s \in \mathbb{R}_+$ also does. Furthermore, (13) implies $\mathbf{a}^T(\bar{\mathbf{x}} - \bar{\mathbf{y}}) \geq 0$. Thus, we can restrict $\mathbf{a}$ to be such that $\mathbf{a}^T(\bar{\mathbf{x}} - \bar{\mathbf{y}}) = 1$. The optimization problem (11) then becomes

$$\max_{\kappa,\mathbf{a}} \kappa \quad \text{s.t.} \quad \frac{1}{\kappa} \geq \sqrt{\mathbf{a}^T\boldsymbol{\Sigma_x}\mathbf{a}} + \sqrt{\mathbf{a}^T\boldsymbol{\Sigma_y}\mathbf{a}} \tag{14}$$

$$\mathbf{a}^T(\bar{\mathbf{x}} - \bar{\mathbf{y}}) = 1,$$

which allows us to eliminate $\kappa$:

$$\min_{\mathbf{a}} \sqrt{\mathbf{a}^T\boldsymbol{\Sigma_x}\mathbf{a}} + \sqrt{\mathbf{a}^T\boldsymbol{\Sigma_y}\mathbf{a}} \quad \text{s.t.} \quad \mathbf{a}^T(\bar{\mathbf{x}} - \bar{\mathbf{y}}) = 1, \tag{15}$$

or, equivalently

$$\min_{\mathbf{a}} \|\mathbf{\Sigma_x}^{1/2}\mathbf{a}\|_2 + \|\mathbf{\Sigma_y}^{1/2}\mathbf{a}\|_2 \quad \text{s.t.} \quad \mathbf{a}^T(\bar{\mathbf{x}} - \bar{\mathbf{y}}) = 1. \tag{16}$$

This is a convex optimization problem, more precisely a second order cone program (SOCP) [8,5]. Furthermore, notice that we can write $\mathbf{a} = \mathbf{a}_0 + \mathbf{Fu}$, where $\mathbf{u} \in \mathbb{R}^{n-1}$, $\mathbf{a}_0 = (\bar{\mathbf{x}} - \bar{\mathbf{y}})/\|\bar{\mathbf{x}} - \bar{\mathbf{y}}\|^2$, and $\mathbf{F} \in \mathbb{R}^{n \times (n-1)}$ is an orthogonal matrix whose columns span the subspace of vectors orthogonal to $\bar{\mathbf{x}} - \bar{\mathbf{y}}$.

Using this we can write (16) as an unconstrained SOCP:

$$\min_{\mathbf{u}} \|\mathbf{\Sigma_x}^{1/2}(\mathbf{a}_0 + \mathbf{Fu})\|_2 + \|\mathbf{\Sigma_y}^{1/2}(\mathbf{a}_0 + \mathbf{Fu})\|_2. \tag{17}$$

We can solve this problem in various ways, for example using interior-point methods for SOCP [8], which yield a worst-case complexity of $O(n^3)$. Of course, the first and second moments of $\mathbf{x}, \mathbf{y}$ must be estimated from data, using for example plug-in estimates $\hat{\mathbf{x}}, \hat{\mathbf{y}}, \hat{\mathbf{\Sigma}}_{\mathbf{x}}, \hat{\mathbf{\Sigma}}_{\mathbf{y}}$ for respectively $\bar{\mathbf{x}}, \bar{\mathbf{y}}, \mathbf{\Sigma_x}, \mathbf{\Sigma_y}$. This brings the total complexity to $O(ln^3)$, where $l$ is the number of data points. This is the same complexity as the quadratic programs one has to solve in support vector machines.

In our implementations, we took an iterative least-squares approach, which is based on the following form, equivalent to (17):

$$\min_{\mathbf{u},\delta,\epsilon} \quad \delta + \frac{1}{\delta}\|\mathbf{\Sigma_x}^{1/2}(\mathbf{a}_0 + \mathbf{Fu})\|_2^2 + \epsilon + \frac{1}{\epsilon}\|\mathbf{\Sigma_y}^{1/2}(\mathbf{a}_0 + \mathbf{Fu})\|_2^2. \tag{18}$$

At iteration $k$, we first minimize with respect to $\delta$ and $\epsilon$ by setting $\delta_k = \|\mathbf{\Sigma_x}^{1/2}(\mathbf{a}_0 + \mathbf{Fu}_{k-1})\|_2$ and $\epsilon_k = \|\mathbf{\Sigma_y}^{1/2}(\mathbf{a}_0 + \mathbf{Fu}_{k-1})\|_2$. Then we minimize with respect to $\mathbf{u}$ by solving a least squares problem in $\mathbf{u}$ for $\delta = \delta_k$ and $\epsilon = \epsilon_k$, which gives us $\mathbf{u}_k$. Because in both update steps the objective of this COP will not increase, the iteration will converge to the global minimum $\|\mathbf{\Sigma_x}^{1/2}(\mathbf{a}_0 + \mathbf{Fu}_*)\|_2 + \|\mathbf{\Sigma_y}^{1/2}(\mathbf{a}_0 + \mathbf{Fu}_*)\|_2$, with $\mathbf{u}_*$ an optimal value of $\mathbf{u}$.

We then obtain $\mathbf{a}_*$ as $\mathbf{a}_0 + \mathbf{Fu}_*$ and $b_*$ from (12) with $\kappa_* = 1/(\sqrt{\mathbf{a}_*^T\mathbf{\Sigma_x}\mathbf{a}_*} + \sqrt{\mathbf{a}_*^T\mathbf{\Sigma_y}\mathbf{a}_*})$. Classification of a new data point $\mathbf{z}_{new}$ is done by evaluating $\text{sign}(\mathbf{a}_*^T\mathbf{z}_{new} - b_*)$: if this is $+1$, $\mathbf{z}_{new}$ is classified as from class $\mathbf{x}$, otherwise $\mathbf{z}_{new}$ is classified as from class $\mathbf{y}$.

It is interesting to see what happens if we make distributional assumptions; in particular, let us assume that $\mathbf{x} \sim \mathcal{N}(\bar{\mathbf{x}}, \mathbf{\Sigma_x})$ and $\mathbf{y} \sim \mathcal{N}(\bar{\mathbf{y}}, \mathbf{\Sigma_y})$. This leads to the following optimization problem:

$$\max_{\alpha,\mathbf{a},b} \alpha \quad \text{s.t.} \quad -b + \mathbf{a}^T\bar{\mathbf{x}} \geq \Phi^{-1}(\alpha)\sqrt{\mathbf{a}^T\mathbf{\Sigma_x}\mathbf{a}} \tag{19}$$

$$b - \mathbf{a}^T\bar{\mathbf{y}} \geq \Phi^{-1}(\alpha)\sqrt{\mathbf{a}^T\mathbf{\Sigma_y}\mathbf{a}}.$$

where $\Phi(z)$ is the cumulative distribution function for a standard normal Gaussian distribution. This has the same form as (8), but now with $\kappa(\alpha) = \Phi^{-1}(\alpha)$ instead of $\kappa(\alpha) = \sqrt{\frac{\alpha}{1-\alpha}}$ (cf. a result by Chernoff [4]). We thus solve the same optimization problem ($\alpha$ disappears from the optimization problem because $\kappa(\alpha)$ is monotone increasing) and find the same decision hyperplane $\mathbf{a}^T z = b$. The difference lies in the value of $\alpha$ associated with $\kappa_*$: $\alpha$ will be higher in this case, so the hyperplane will have a higher predicted probability of classifying future data correctly.

## 3 Kernelization

In this section we describe the "kernelization" of the minimax approach described in the previous section. We seek to map the problem to a higher dimensional feature space $\mathbb{R}^f$ via a mapping $\varphi : \mathbb{R}^n \mapsto \mathbb{R}^f$, such that a linear discriminant in the feature space corresponds to a nonlinear discriminant in the original space. To carry out this programme, we need to try to reformulate the minimax problem in terms of a kernel function $K(\mathbf{z}_1, \mathbf{z}_2) = \varphi(\mathbf{z}_1)^T \varphi(\mathbf{z}_2)$ satisfying Mercer's condition.

Let the data be mapped as $\mathbf{x} \mapsto \varphi(\mathbf{x}) \sim (\overline{\varphi(\mathbf{x})}, \mathbf{\Sigma}_{\varphi(\mathbf{x})})$ and $\mathbf{y} \mapsto \varphi(\mathbf{y}) \sim (\overline{\varphi(\mathbf{y})}, \mathbf{\Sigma}_{\varphi(\mathbf{y})})$ where $\{\mathbf{x}_i\}_{i=1}^{N_x}$ and $\{\mathbf{y}_i\}_{i=1}^{N_y}$ are training data points in the classes corresponding to $\mathbf{x}$ and $\mathbf{y}$ respectively. The decision hyperplane in $\mathbb{R}^f$ is then given by $\mathbf{a}^T \varphi(\mathbf{z}) = b$ with $\mathbf{a}, \varphi(\mathbf{z}) \in \mathbb{R}^f$ and $b \in \mathbb{R}$. In $\mathbb{R}^f$, we need to solve the following optimization problem:

$$\min_{\mathbf{a}} \sqrt{\mathbf{a}^T \mathbf{\Sigma}_{\varphi(\mathbf{x})} \mathbf{a}} + \sqrt{\mathbf{a}^T \mathbf{\Sigma}_{\varphi(\mathbf{y})} \mathbf{a}} \quad \text{s.t.} \quad \mathbf{a}^T (\overline{\varphi(\mathbf{x})} - \overline{\varphi(\mathbf{y})}) = 1, \qquad (20)$$

where, as in (12), the optimal value of $b$ will be given by

$$b_* = \mathbf{a}_*^T \overline{\varphi(\mathbf{x})} - \kappa_* \sqrt{\mathbf{a}_*^T \mathbf{\Sigma}_{\varphi(\mathbf{x})} \mathbf{a}_*} = \mathbf{a}_*^T \overline{\varphi(\mathbf{y})} + \kappa_* \sqrt{\mathbf{a}_*^T \mathbf{\Sigma}_{\varphi(\mathbf{y})} \mathbf{a}_*}, \qquad (21)$$

where $\mathbf{a}_*$ and $\kappa_*$ are the optimal values of $\mathbf{a}$ and $\kappa$ respectively. However, we do not wish to solve the COP in this form, because we want to avoid using $f$ or $\varphi$ explicitly.

If $\mathbf{a}$ has a component in $\mathbb{R}^f$ which is orthogonal to the subspace spanned by $\varphi(\mathbf{x}_i)$, $i = 1, 2, \ldots, N_x$ and $\varphi(\mathbf{y}_i)$, $i = 1, 2, \ldots, N_y$, then that component won't affect the objective or the constraint in (20). This implies that we can write $\mathbf{a}$ as

$$\mathbf{a} = \sum_{i=1}^{N_x} \alpha_i \varphi(\mathbf{x}_i) + \sum_{j=1}^{N_y} \beta_j \varphi(\mathbf{y}_j). \qquad (22)$$

Substituting expression (22) for $\mathbf{a}$ and estimates $\widehat{\varphi(\mathbf{x})} = \frac{1}{N_x} \sum_{i=1}^{N_x} \varphi(\mathbf{x}_i)$ , $\widehat{\varphi(\mathbf{y})} = \frac{1}{N_y} \sum_{i=1}^{N_y} \varphi(\mathbf{y}_i)$, $\hat{\mathbf{\Sigma}}_{\varphi(\mathbf{x})} = \frac{1}{N_x} \sum_{i=1}^{N_x} (\varphi(\mathbf{x}_i) - \widehat{\varphi(\mathbf{x})})(\varphi(\mathbf{x}_i) - \widehat{\varphi(\mathbf{x})})^T$ and $\hat{\mathbf{\Sigma}}_{\varphi(\mathbf{y})} = \frac{1}{N_y} \sum_{i=1}^{N_y} (\varphi(\mathbf{y}_i) - \widehat{\varphi(\mathbf{y})})(\varphi(\mathbf{y}_i) - \widehat{\varphi(\mathbf{y})})^T$ for the means and the covariance matrices in the objective and the constraint of the optimization problem (20), we see that both the objective and the constraints can be written in terms of the kernel function $K(\mathbf{z}_1, \mathbf{z}_2) = \varphi(\mathbf{z}_1)^T \varphi(\mathbf{z}_2)$. We obtain:

$$\min_{\gamma} \sqrt{\frac{1}{N_x} \gamma^T \tilde{\mathbf{K}}_\mathbf{x}^T \tilde{\mathbf{K}}_\mathbf{x} \gamma} + \sqrt{\frac{1}{N_y} \gamma^T \tilde{\mathbf{K}}_\mathbf{y}^T \tilde{\mathbf{K}}_\mathbf{y} \gamma} \quad \text{s.t.} \quad \gamma^T (\tilde{\mathbf{k}}_\mathbf{x} - \tilde{\mathbf{k}}_\mathbf{y}) = 1, \qquad (23)$$

where $\gamma = [\alpha_1 \; \alpha_2 \; \cdots \; \alpha_{N_x} \; \beta_1 \; \beta_2 \; \cdots \; \beta_{N_y}]^T$, $\tilde{\mathbf{k}}_\mathbf{x} \in \mathbb{R}^{N_x + N_y}$ with $[\tilde{\mathbf{k}}_\mathbf{x}]_i = \frac{1}{N_x} \sum_{j=1}^{N_x} K(\mathbf{x}_j, \mathbf{z}_i)$, $\tilde{\mathbf{k}}_\mathbf{y} \in \mathbb{R}^{N_x + N_y}$ with $[\tilde{\mathbf{k}}_\mathbf{y}]_i = \frac{1}{N_y} \sum_{j=1}^{N_y} K(\mathbf{y}_j, \mathbf{z}_i)$, $\mathbf{z}_i = \mathbf{x}_i$ for $i = 1, 2, \ldots, N_x$ and $\mathbf{z}_i = \mathbf{y}_{i - N_x}$ for $i = N_x + 1, N_x + 2, \ldots, N_x + N_y$. $\tilde{\mathbf{K}}$ is defined as:

$$\tilde{\mathbf{K}} = \begin{pmatrix} \mathbf{K}_\mathbf{x} - \mathbf{1}_{N_x} \tilde{\mathbf{k}}_\mathbf{x}^T \\ \mathbf{K}_\mathbf{y} - \mathbf{1}_{N_y} \tilde{\mathbf{k}}_\mathbf{y}^T \end{pmatrix} = \begin{pmatrix} \tilde{\mathbf{K}}_\mathbf{x} \\ \tilde{\mathbf{K}}_\mathbf{y} \end{pmatrix} \qquad (24)$$

where $\mathbf{1}_m$ is a column vector with ones of dimension $m$. $\mathbf{K}_\mathbf{x}$ and $\mathbf{K}_\mathbf{y}$ contain respectively the first $N_x$ rows and the last $N_y$ rows of the Gram matrix $\mathbf{K}$ (defined as $\mathbf{K}_{ij} = \varphi(\mathbf{z}_i)^T \varphi(\mathbf{z}_j) = K(\mathbf{z}_i, \mathbf{z}_j)$). We can also write (23) as

$$\min_{\gamma} \|\frac{\tilde{\mathbf{K}}_\mathbf{x}}{\sqrt{N_x}} \gamma\|_2 + \|\frac{\tilde{\mathbf{K}}_\mathbf{y}}{\sqrt{N_y}} \gamma\|_2 \quad \text{s.t.} \quad \gamma^T (\tilde{\mathbf{k}}_\mathbf{x} - \tilde{\mathbf{k}}_\mathbf{y}) = 1, \qquad (25)$$

which is a second order cone program (SOCP) [5] that has the same form as the SOCP in (16) and can thus be solved in a similar way. Notice that, in this case, the optimizing variable is $\gamma \in \mathbb{R}^{N_x + N_y}$ instead of $\mathbf{a} \in \mathbb{R}^n$. Thus the dimension of the optimization problem increases, but the solution is more powerful because the kernelization corresponds to a more complex decision boundary in $\mathbb{R}^n$.

Similarly, the optimal value $b_*$ of $b$ in (21) will then become

$$b_* = \gamma_*^T \tilde{\mathbf{k}}_\mathbf{x} - \kappa_* \sqrt{\frac{1}{N_x} \gamma_*^T \tilde{\mathbf{K}}_\mathbf{x}^T \tilde{\mathbf{K}}_\mathbf{x} \gamma_*} = \gamma_*^T \tilde{\mathbf{k}}_\mathbf{y} + \kappa_* \sqrt{\frac{1}{N_y} \gamma_*^T \tilde{\mathbf{K}}_\mathbf{y}^T \tilde{\mathbf{K}}_\mathbf{y} \gamma_*}, \quad (26)$$

where $\gamma_*$ and $\kappa_*$ are the optimal values of $\gamma$ and $\kappa$ respectively.

Once $\gamma_*$ is known, we get $\kappa_* = 1/\left( \sqrt{\frac{1}{N_x} \gamma_*^T \tilde{\mathbf{K}}_\mathbf{x}^T \tilde{\mathbf{K}}_\mathbf{x} \gamma_*} + \sqrt{\frac{1}{N_y} \gamma_*^T \tilde{\mathbf{K}}_\mathbf{y}^T \tilde{\mathbf{K}}_\mathbf{y} \gamma_*} \right)$ and then $b_*$ from (26). Classification of a new data point $\mathbf{z}_{new}$ is then done by evaluating $\text{sign}(\mathbf{a}_*^T \varphi(\mathbf{z}_{new}) - b_*) = \text{sign}\left( \left( \sum_{i=1}^{N_x + N_y} [\gamma_*]_i K(\mathbf{z}_i, \mathbf{z}_{new}) \right) - b_* \right)$ (again only in terms of the kernel function): if this is $+1$, $\mathbf{z}_{new}$ is classified as from class $\mathbf{x}$, otherwise $\mathbf{z}_{new}$ is classified as from class $\mathbf{y}$.

# 4   Experiments

In this section we report the results of experiments that we carried out to test our algorithmic approach. The validity of $1 - \alpha$ as the worst case bound on the probability of misclassification of future data is checked, and we also assess the usefulness of the kernel trick in this setting. We compare linear kernels and Gaussian kernels.

Experimental results on standard benchmark problems are summarized in Table 1. The Wisconsin breast cancer dataset contained 16 missing examples which were not used. The breast cancer, pima, diabetes, ionosphere and sonar data were obtained from the UCI repository. Data for the twonorm problem data were generated as specified in [3]. Each dataset was randomly partitioned into 90% training and 10% test sets. The kernel parameter ($\sigma$) for the Gaussian kernel ($e^{-\|x-y\|^2/\sigma}$) was tuned using cross-validation over 20 random partitions. The reported results are the averages over 50 random partitions for both the linear kernel and the Gaussian kernel with $\sigma$ chosen as above.

The results are comparable with those in the existing literature [3] and with those obtained with Support Vector Machines. Also, we notice that $\alpha$ is indeed smaller

Table 1: $\alpha$ and test-set accuracy (TSA) compared to BPB (best performance in [3]) and to the performance of an SVM with linear kernel (SVML) and an SVM with Gaussian kernel (SVMG)

| Dataset | Linear kernel | | Gaussian kernel | | BPB | SVML | SVMG |
|---|---|---|---|---|---|---|---|
| | $\alpha$ | TSA | $\alpha$ | TSA | | | |
| Twonorm | 80.2 % | 96.0 % | 83.6 % | 97.2 % | 96.3 % | 95.6 % | 97.4 % |
| Breast cancer | 84.4 % | 97.2 % | 92.7 % | 97.3 % | 96.8 % | 92.6 % | 98.5 % |
| Ionosphere | 63.3 % | 85.4 % | 89.9 % | 93.0 % | 93.7 % | 87.8 % | 91.5 % |
| Pima diabetes | 31.2 % | 73.8 % | 33.0 % | 74.6 % | 76.1 % | 70.1 % | 75.3 % |
| Sonar | 62.4 % | 75.1 % | 87.1 % | 89.8 % | - | 75.9 % | 86.7 % |

than the test-set accuracy in all cases. Furthermore, $\alpha$ is smaller for a linear decision boundary then for the nonlinear decision boundary obtained via the Gaussian kernel. This clearly shows that kernelizing the method leads to more powerful decision boundaries.

## 5    Conclusions

The problem of linear discrimination has a long and distinguished history. Many results on misclassification rates have been obtained by making distributional assumptions (e.g., Anderson and Bahadur [1]). Our results, on the other hand, make use of recent work on moment problems and semidefinite optimization to obtain distribution-free results for linear discriminants. We have also shown how to exploit Mercer kernels to generalize our algorithm to nonlinear classification.

The computational complexity of our method is comparable to the quadratic program that one has to solve for the support vector machine (SVM). While we have used a simple iterative least-squares approach, we believe that there is much to gain from exploiting analogies to the SVM and developing specialized, more efficient optimization procedures for our algorithm, in particular tools that break the data into subsets. The extension towards large scale applications is a current focus of our research, as is the problem of developing a variant of our algorithm for multiway classification and function regression. Also the statistical consequences of using plug-in estimates for the mean vectors and covariance matrices needs to be investigated.

### Acknowledgements

We would like to acknowledge support from ONR MURI N00014-00-1-0637, from NSF grants IIS-9988642 and ECS-9983874 and from the Belgian American Educational Foundation.

## References

[1] Anderson, T. W. and Bahadur, R. R. (1962) Classification into two multivariate Normal distributions with different covariance matrices. *Annals of Mathematical Statistics* **33**(2): 420-431.

[2] Bertsimas, D. and Sethuraman, J. (2000) Moment problems and semidefinite optimization. *Handbook of Semidefinite Optimization* 469-509, Kluwer Academic Publishers.

[3] Breiman L. (1996) Arcing classifiers. Technical Report 460, Statistics Department, University of California, December 1997.

[4] Chernoff H. (1972) The selection of effective attributes for deciding between hypothesis using linear discriminant functions. In *Frontiers of Pattern Recognition*, (S. Watanabe, ed.), 55-60. New York: Academic Press.

[5] Boyd, S. and Vandenberghe, L. (2001) *Convex Optimization*. Course notes for EE364, Stanford University. Available at `http://www.stanford.edu/class/ee364`.

[6] Isii, K. (1963) On the sharpness of Chebyshev-type inequalities. *Ann. Inst. Stat. Math.* **14**: 185-197.

[7] Mika, M. Rätsch, G., Weston, J., Schölkopf, B., and Müller, K.-R. (1999) Fisher discriminant analysis with kernels. In *Neural Networks for Signal Processing IX*, 41–48, New York: IEEE Press.

[8] Nesterov, Y. and Nemirovsky, A. (1994) *Interior Point Polynomial Methods in Convex Programming: Theory and Applications*. Philadelphia, PA: *SIAM*.
